# Sampling Techniques for Kernel Methods

**Dimitris Achlioptas**
Microsoft Research
*optas@microsoft.com*

**Frank McSherry**
University of Washington
*mcsherry@cs.washington.edu*

**Bernhard Schölkopf**
Biowulf Technologies NY
*bs@conclu.de*

## Abstract

We propose randomized techniques for speeding up Kernel Principal Component Analysis on three levels: sampling and quantization of the Gram matrix in training, randomized rounding in evaluating the kernel expansions, and random projections in evaluating the kernel itself. In all three cases, we give sharp bounds on the accuracy of the obtained approximations. Rather intriguingly, all three techniques can be viewed as instantiations of the following idea: replace the kernel function $k$ by a "randomized kernel" which behaves like $k$ in expectation.

## 1 Introduction

Given a collection $\mathcal{X}$ of training data $x_1, \ldots, x_m$, techniques such as linear SVMs [13] and PCA extract features from $\mathcal{X}$ by computing linear functions of this data. However, it is often the case that the structure present in the training data is not simply a linear function of the data representation. Worse, many data sets do not readily support linear operations such as addition and scalar multiplication (text, for example).

In a "kernel method" $\mathcal{X}$ is first mapped into some dot product space $\mathcal{H}$ using $\Phi : \mathcal{X} \to \mathcal{H}$. The dimension of $\mathcal{H}$ can be very large, even infinite, and therefore it may not be practical (or possible) to work with the mapped data explicitly. Nonetheless, in many cases the dot products $\langle \Phi(x_i), \Phi(x_j) \rangle$ can be evaluated efficiently using a positive definite kernel $k$ for $\Phi$, i.e. a function $k$ so that $k(x_i, x_j) = \langle \Phi(x_i), \Phi(x_j) \rangle$.

Any algorithm whose operations can be expressed in terms of dot products can be generalized to an algorithm which operates on $\Phi(\mathcal{X})$, simply by presenting the *Gram* matrix

$$K_{ij} := k(x_i, x_j)$$

as the input covariance matrix. Note that at no point is the function $\Phi$ explicitly computed; the kernel $k$ implicitly performs the dot product calculations between mapped points.

While this "kernel trick" has been extremely successful, a problem common to all kernel methods is that, in general, $K$ is a dense matrix, making the input size scale as $m^2$. For example, in Kernel PCA such a matrix has to be diagonalized, while in SVMs a quadratic program of size $m^2$ must be solved. As the size of training sets in practical applications increases, the growth of the input size rapidly poses severe computational limitations.

Various methods have been proposed to deal with this issue, such as decomposition methods for SVM training (e.g., [10]), speedup methods for Kernel PCA [12], and other kernel methods [2, 14]. Our research is motivated by the need for such speedups that are also accompanied by strong, provable performance guarantees.

In this paper we give three such speedups for Kernel PCA. We start by simplifying the Gram matrix via a novel matrix sampling/quantization scheme, motivated by spectral properties of random matrices. We then move on to speeding up classification, by using randomized rounding in evaluating kernel expansions. Finally, we consider the evaluation of kernel functions themselves and show how many popular kernels can be approximated efficiently.

Our first technique relates matrix simplification to the stability of invariant subspaces. The other two are, in fact, completely general and apply to all kernel methods. What is more, our techniques suggest the notion of *randomized kernels*, whereby each evaluation of the kernel $k$ is replaced by an evaluation of a randomized function $\widehat{k}$ (on the same input pair). The idea is to use a function $\widehat{k}$ which for every input pair behaves like $k$ in expectation (over its internal coin-flips), yet confers significant computational benefits compared to using $k$. In fact, each one of our three techniques can be readily cast as an appropriate randomized kernel, with no other intervention.

## 2   Kernel PCA

Given $m$ training points recall that $K$ is an $m \times m$ matrix with $K_{ij} = k(x_i, x_j)$. For some choice of $\ell \leq m$, the Kernel PCA (KPCA) method [11] computes the $\ell$ largest eigenvalues, $\lambda_1, \ldots, \lambda_\ell$, and eigenvectors, $\alpha^1, \ldots, \alpha^\ell$ of $K$. Then, given an input point $x$, the method computes the value of $\ell$ nonlinear feature extraction functions

$$f_n(x) = \lambda_n^{-1/2} \sum_{i=1}^m \alpha_i^n k(x_i, x) \ \ .$$

There are several methods for computing the principal components of a symmetric matrix. The choice depends on the properties of the matrix and on how many components one is seeking. In particular, if relatively few principal components are required, as is the case in KPCA, *Orthogonal Iteration* is a commonly used method.[1]

> **Orthogonal Iteration**$(A, \ell)$
>
> 1. Let $Q$ be a random $m \times \ell$ matrix with orthonormal columns.
> 2. While not converged, do
>    - (a)  $Q \leftarrow AQ$
>    - (b)  $Q \leftarrow \text{Orthonormalize}(Q)$
> 3. Return $Q$

It is worth looking closely at the complexity of performing Orthogonal Iteration on a matrix $A$. Step 1 can be done in $O(m\ell)$ steps, making step 2 the computational bottleneck. The orthonormalization step 2b takes time $O(m\ell^2)$ and is overwhelmed by the cost of computing $AQ$ in step 2a which, generally, takes $O(m^2\ell)$. The number of iterations of the while loop is a somewhat complicated issue, but one can prove that the "error" in $Q$ (with respect to the true principal components) decreases exponentially with the number of iterations. All in all, the running time of Orthogonal Iteration scales linearly with the cost of the matrix multiplication $AQ$. If $A$ is sparse, i.e., if roughly one out of every $s$ entries of $A$ is non-zero, then the matrix multiplication $AQ$ costs $O(m^2\ell/s)$.

As mentioned earlier, the matrix $K$ used in Kernel PCA is almost never sparse. In the next section, we will show how to sample and quantize the entries of $K$, obtaining a matrix $\widehat{K}$ which is sparser and whose entries have simpler data representation, yet has essentially the same spectral structure, i.e. eigenvalues/eigenvectors, as $K$.

# 3  Sampling Gram Matrices

In this section we describe two general "matrix simplification" techniques and discuss their implications for Kernel PCA. In particular, under natural assumptions on the spectral structure of $K$, we will prove that applying KPCA to the simplified matrix $\widehat{K}$ yields subspaces which are very close to those that KPCA would find in $K$. As a result, when we project vectors onto these spaces (as performed by the feature extractors) the results are provably close to the original ones.

First, our sparsification process works by randomly omitting entries in $K$. Precisely stated, we let the matrix $\widehat{K}$ be described entrywise as

$$
\widehat{K}_{ij} \;=\; \widehat{K}_{ji} \;=\; \begin{cases} 0 & \text{with probability } 1 - 1/s \\[2mm] sK_{ij} & \text{with probability } 1/s \;. \end{cases}
$$

Second, our quantization process rounds each entry in $K$ to one of $\{-b, +b\}$, where $b = \max_{i,j} |K_{ij}|$, thus reducing the representation of each entry to a single bit.

$$
\widehat{K}_{ij} \;=\; \widehat{K}_{ji} \;=\; \begin{cases} +b & \text{with probability } 1/2 + K_{ij}/(2b) \\[2mm] -b & \text{with probability } 1/2 - K_{ij}/(2b) \;. \end{cases}
$$

Sparsification greatly accelerates the computation of eigenvectors by accelerating multiplication by $\widehat{K}$. Moreover, both approaches greatly reduce the space required to store the matrix (and they can be readily combined), allowing for much bigger training sets to fit in main memory. Finally, we note that i) sampling also speeds up the construction of the Gram matrix since we need only compute those values of $K$ that remain in $\widehat{K}$, while ii) quantization allows us to replace exact kernel evaluations by coarse unbiased estimators, which can be more efficient to compute.

While the two processes above are quite different, they share one important commonality: in each case, $\mathbf{E}(\widehat{K}_{ij}) = K_{ij}$. Moreover, the entries of the error matrix, $E_K = \widehat{K} - K$, are independent random variables, having expectation zero and bounded variance. Large deviation extensions [5] of Wigner's famous semi-circle law, imply that with very high probability such matrices have small L2 norm (denoted by $\| \cdot \|$ throughout).

**Theorem 1 (Furedi and Komlos [5])** *Let $E_K$ be an $m \times m$ symmetric matrix whose entries are independent random variables with mean 0, variance bounded above by $\sigma^2$, and magnitude bounded by $\sigma\sqrt{m}/\log^3 m$. With probability $1 - 2\exp(-\sigma^2 m/8)$,*

$$
\|E_K\| \;\leq\; 4\sigma\sqrt{m} \;.
$$

It is worth noting that this upper bound is within a constant factor of the *lower* bound on the L2 norm of *any* matrix where the mean squared entry equals $\sigma^2$. More precisely, it is easy to show that every matrix with Frobenius norm $(\sigma m)^2$ has L2 norm at least $\sigma\sqrt{m}$. Therefore, we see that the L2 error introduced by $\widehat{K}$ is within a factor of 4 of the L2 error associated with *any* modification to $K$ that has the same entrywise mean squared error.

We will analyze three different cases of spectral stability, corresponding to progressively stronger assumptions. At the heart of these results is the stability of invariant subspaces in the presence of additive noise. This stability is very strong, but can be rather technical to express. In stating each of these results, it is important to note that the eigenvectors correspond exactly to the feature extractors associated with Kernel PCA. For an input point $x$, let $v$ denote the vector whose $i$th coordinate is $k(x_i, x)$ and recall that

$$
f_n(x) \;=\; \lambda_n^{-1/2} \sum_{i=1}^m \alpha_i^n v_i \;\sim\; \langle \alpha^n, v \rangle \;.
$$

Recall that in KPCA we associate features with the $\ell$ largest eigenvalues of $\widehat{K}$, where $\ell$ is typically chosen by requiring $\lambda_\ell \geq t > \lambda_{\ell+1}$, for some threshold $t$. First, we consider what happens when $\lambda_\ell - \lambda_{\ell+1}$ is not large. Observe that in this case we cannot hope to equate all $f_n(x)$ and $\widehat{f}_n(x)$, as the $\ell$th feature is very sensitive to small changes in $\lambda_\ell$. However, we can show that all features with $\lambda_n$ far from $t$ are treated consistently in $K$ and $\widehat{K}$.

**Theorem 2** *Let $F(t)$ be any matrix whose columns form an orthonormal basis for the space of features (eigenvectors) in $K$ whose eigenvalue is at least $t$. Let $F_\perp(t)$ be any matrix whose columns form an orthonormal basis for the orthogonal complement of $F(t)$. Let $\widehat{F}(t)$ and $\widehat{F}_\perp(t)$ be the analogous matrices for $\widehat{K}$. For any $b > 0$,*

$$\|\widehat{F}^T(t)\ F_\perp(t-b)\| \leq \frac{\|E_K\|}{b} \quad and \quad \|F^T(t+b)\ \widehat{F}_\perp(t)\| \leq \frac{\|E_K\|}{b} \ .$$

If we use the threshold $t$ for the eigenvalues of $\widehat{K}$, the first equation asserts that the features KPCA recovers are not among the features of $K$ whose eigenvalues are less than $t - b$. Similarly, the second equation asserts that KPCA will recover all the features of $K$ whose eigenvalues are larger than $t + b$.

*Proof:* We employ the techniques of Davis and Kahan [4]. Observe that

$$\widehat{K} - K = E_K$$
$$\widehat{F}^T(t)\widehat{K}F_\perp(t-b) - \widehat{F}^T(t)KF_\perp(t-b) = \widehat{F}^T(t)E_K F_\perp(t-b)$$
$$\widehat{D}\widehat{F}^T(t)F_\perp(t-b) - \widehat{F}^T(t)F_\perp(t-b)D = \widehat{F}^T(t)E_K F_\perp(t-b)$$

where $\widehat{D}$ and $D$ are diagonal matrices whose entries (the eigenvalues of $\widehat{K}$ and $K$) are at least $t$ and at most $t - b$, respectively. Therefore

$$t\|\widehat{F}^T(t)F_\perp(t-b)\| - \|\widehat{F}^T(t)F_\perp(t-b)\|(t-b) \leq \|\widehat{F}^T(t)E_K F_\perp(t-b)\|$$
$$b\|\widehat{F}^T(t)F_\perp(t-b)\| \leq \|E_K\|$$

which implies the first stated result. The second proof is essentially identical. $\square$

In our second result we will still not be able to isolate individual features, as the error matrix can reorder their importance by, say, interchanging $f_n$ and $f_{n+1}$. However, we can show that any such interchange will occur consistently in all test vectors. Let $f(x)$ be the $\ell$-dimensional vector whose $n$th coordinate is $\lambda_n^{1/2} f_n(x)$, i.e., here we do *not* normalize features to "equal importance". Recall that $v$ is the vector whose $i$th coordinate is $k(x_i, x)$.

**Theorem 3** *Assume that $\lambda_\ell - \lambda_{\ell+1} \geq 2c\|E_K\|$ for some $c \geq 1$. There is an orthonormal rotation matrix $R$ such that for all $x$*

$$\|\widehat{f}(x) - R \cdot f(x)\| \leq \|v\|/c \ .$$

*Proof:* Instantiate Theorem 2 with $b = 2c\|E_K\|$ and $t = \lambda_\ell$. $\square$

Note that the rotation matrix becomes completely irrelevant if we are only concerned with differences, angles, or inner products of feature vectors.

Finally, we prove that in the special case where a feature is well separated from its neighboring features in the spectrum of $K$, we get a particularly strong bound.

**Theorem 4** *If $c \geq 1$, $\lambda_n - \lambda_{n+1} > 2c\|E_K\|$, and $\lambda_{n-1} - \lambda_n > 2c\|E_K\|$, then*

$$|f_n(x) - \widehat{f}_n(x)| \leq \|v\|/c \ .$$

*Proof:(sketch)* As before, we specialize Theorem 2, but first shift both $K$ and $\widehat{K}$ by $\lambda_n I_m$. This does not change the eigenvectors, and allows us to consider $f_n$ in isolation. $\square$

## 4   Approximating Feature Extractors Quickly

Having determined eigenvalues and eigenvectors, given an input point $x$, the value of $x$ on each feature reduces to evaluating, for some unit vector $\alpha$, a function

$$f(x) = \sum_{i=1}^{m} \alpha_i k(x, x_i) \ ,$$

where we dropped the subscript $n$, as well as the scaling by $\lambda_n^{-1/2}$. Assume that $k(x, \cdot)$ take values in an interval of width $c$ and let $||\alpha||$ be any unit vector. We will devise a fast, unbiased, small-variance estimator for $f$, by sampling and rounding the expansion coefficients $\alpha_i$.

Fix $s > 0$. For each $i$: if $|\alpha_i| \geq 1/s$ then let $\widehat{\alpha}_i = s\alpha_i$; if $|\alpha_i| < 1/s$ let

$$\widehat{\alpha}_i \ = \ \begin{cases} \mathrm{sgn}(\alpha_i) & \text{with probability } s \cdot |\alpha_i| \\[2mm] 0 & \text{otherwise.} \end{cases}$$

That is, after potentially keeping some large coefficients deterministically, we proceed to perform "randomized rounding" on the (remaining) coefficients of $\alpha$. Let

$$\widehat{f}(x) = s^{-1} \sum_{i=1}^{m} \widehat{\alpha}_i k(x, x_i) \ .$$

Clearly, we have $\mathbf{E}[\widehat{f}(x)] = f(x)$. Moreover, using Hoeffding's inequality [7], we can bound the behavior of $|f(x) - \widehat{f}(x)|$ arising from the terms subjected to probabilistic rounding. In particular, this gives

$$\Pr\left[ |f(x) - \widehat{f}(x)| \geq t \right] \leq 2 \exp\left( -\frac{2(st)^2}{mc^2} \right) \ . \tag{1}$$

Note now that in Kernel PCA we typically expect $\alpha_i \approx 1/\sqrt{m}$, i.e., dense eigenvectors. This makes $c\sqrt{m}$ the natural scale for measuring $f(x)$ and suggests that using far fewer than $m$ kernel evaluations we can get good approximations of $f(x)$. In particular, for a chosen (fixed) value of $T$ let us say that $f_j(x)$ is *trivial* if

$$f_j(x) < T \times \sqrt{\log m} \ .$$

Having picked some threshold $T$ (for SVM expansions $T$ is related to the classification offset) we want to determine whether $f(x)$ is non-trivial and, if so, we want to get a good *relative* error estimate for it.

**Theorem 5** *For any $\epsilon \in (0, 2/3)$ and $T \leq (c/\epsilon)\sqrt{m/\log m}$ set $s = 2c\sqrt{m}/(\epsilon T)$. With probability at least $1 - 1/m^3$*

1. *There are fewer than $4c\epsilon^{-1} \times \dfrac{m}{T}$ non-zero $\widehat{\alpha}_i$.*

2. *Either both $f_j(x)$ and $\widehat{f}_j(x)$ are trivial or*
$$(1 - \epsilon) f_j(x) \ \leq \ \widehat{f}_j(x) \ \leq \ (1 + \epsilon) f_j(x) \ .$$

*Proof:* Let $\widehat{m}$ denote the number of non-zero $\widehat{\alpha}_i$ and let $I = \{i : |\alpha_i| \geq 1/s\}$. Note that $\widehat{m}$ equals $|I|$ plus the sum of $m - |I|$ independent Bernoulli trials. It is not hard to show that the probability that the event in 1 fails is bounded by the corresponding probability for the case where all coordinates of $||\alpha||$ are equal. In that case, $\widehat{m}$ is a Binomial random variable with $m$ trials and probability of success $s/\sqrt{m}$ and, by our choice of $s$, $\mathbf{E}(\widehat{m}) = 2cm/(\epsilon T)$. The Chernoff bound now implies that the event in 1 fails to occur with probability $o(m^{-3})$. For the enent in 2 it suffices to observe that failure occurs only if $|\widehat{f}(x) - f(x)|$ is at least $(2\epsilon/3)T\sqrt{\log m}$. By (1), this also occurs with probability $o(m^{-3})$. $\qquad\square$

# 5 Quick batch approximations of Kernels

In this section we devise fast approximations of the kernel function itself. We focus on kernels sharing the following two characteristics: i) they map $d$-dimensional Euclidean space, and, ii) the mapping depends only on the distance and/or inner product of the considered points. We note that this covers some of the most popular kernels, e.g., RBFs and polynomial kernels. To simplify exposition we focus on the following task: given a sequence of (test) vectors $x_1, x_2, \ldots$, determine $k(x_i, y_j)$ for each of a fixed set of (training) vectors $y_1, \ldots, y_m$, where $m \gg d$.

To get a fast batch approximdition, the idea is that rather than evaluating distances and inner products directly, we will use a fast, approximately correct oracle for these quantities offering the following guarantee: it will answer *all* queries with small *relative* error.

A natural approach for creating such an oracle is to pick $s$ of the $d$ coordinates in input space and use the projection onto these coordinates to determine distances and inner products. The problem with this approach is that if $x_i - y_j = (0, \ldots, 0, \|x_i - y_j\|, 0, \ldots, 0)$, any coordinate sampling scheme is bound to do poorly. On the other hand, if we knew that all coordinates contributed "approximately equally" to $k(x_i, y_j)$, then coordinate sampling would be much more appealing. We will do just this, using the technique of *random projections* [8], which can be viewed as coordinate sampling preceded by a random rotation.

Imagine that we applied a spherically random rotation $R$ to $y_1, \ldots, y_m$ (before training) and then applied the same random rotation $R$ to each input point $x_i$ as it became available. Clearly, all distances and inner products would remain the same and we would get exactly the same results as without the rotation. The interesting part is that *any* fixed vector $z$ that was a linear combination of training and/or input vectors, e.g. $x_i - y_j$, after being rotated becomes a spherically *random* vector of length $\|z\|$. As a result, the coordinates of $z$ are i.i.d. random variables, in fact $N(0, \|z\|/\sqrt{d})$, enabling coordinate sampling.

Our oracle amounts to multiplying each training and input point by the same $d \times s$ projection matrix $P$, where $s = O(\log d)$, and using the resulting $s$-dimensional points to estimate distances and inner products. (Think of $P$ as the result of taking a $d \times d$ rotation matrix $R$ and keeping the first $s$ columns (sampling)). Before describing the choice of $P$ and the quality of the resulting approximations, let us go over the computational savings.

1. Rotating the $m$ training vectors takes $O(md \log d)$. Note that
   - This cost will be amortized over the sequence of input vectors.
   - This rotation can be performed in the training phase.
2. The kernel evaluations for each $x_i$ now take $O(m \log d)$ instead of $O(md)$.
3. Rotating $x$ takes time $O(d \log d)$ which is dominated by $O(m \log d)$.

Having motivated our oracle as a spherically random rotation followed by coordinate sampling, we will actually employ a simpler method to perform the projection. Namely, we will rely on a recent result of [1], asserting that we can do at least as well by taking $P_{ij} = p_{ij}/\sqrt{d}$, where the $\{p_{ij}\}$ are i.i.d. with $p_{ij} \in \{-1, +1\}$, each case having probability $1/2$. Thus, postponing the scaling by $1/\sqrt{d}$ until the end, each of the $s$ new coordinates is formed as follows: split the $d$ coordinates randomly into two groups; sum the coordinates in each group; take the difference of the two sums.

Regarding the quality of approximations we get

**Theorem 6** *Consider any sets of points $y_1, \ldots, y_m$ and $x_1, \ldots, x_t$ in $\mathbb{R}^d$. Let $n = m + t$ and for given $\beta, \epsilon > 0$ let*

$$s = \frac{4 + 2\beta}{\epsilon^2/2 - \epsilon^3/3} \ln n \ .$$

*Let $P$ be a random $d \times s$ matrix defined by $P_{ij} = p_{ij}/\sqrt{d}$, where the $\{p_{ij}\}$ are i.i.d. with $p_{ij} \in \{-1, +1\}$, each case having probability $1/2$. For any $z \in \mathbb{R}^d$ let $\widehat{z}$ denote $zP$.*

*With probability at least $1 - 1/n^{\beta}$, for* every *pair of points $x_i, y_j$*

$$(1 - \epsilon)\|x_i - y_j\|^2 \le \|\widehat{x}_i - \widehat{y}_j\|^2 \le (1 + \epsilon)\|x_i - y_j\|^2 \tag{2}$$

*and*

$$|\langle \widehat{x}_i \widehat{y}_j \rangle - \langle x_i y_j \rangle| < \epsilon\|x_i\|^2 + \epsilon\|y_j\|^2 \ . \tag{3}$$

*Proof:* We use Lemma 5 of [1], asserting that for any $z \in \mathbb{R}^d$ and any $\epsilon > 0$,

$$\Pr\left[(1 - \epsilon)\|z\|^2 \le \|\widehat{z}\|^2 \le (1 + \epsilon)\|z\|^2\right] > 1 - 2\exp\left(-\frac{s}{2}(\epsilon^2/2 - \epsilon^3/3)\right) \ . \tag{4}$$

By our choice of $s$, the r.h.s. of (4) is $1 - 2/n^{2+\beta}$. Thus, by the union bound, with probability at least $1 - 1/n^{\beta}$ the lengths of all $mt + m + t \le \binom{n}{2}$ vectors corresponding to $x_i - y_j$ and $x_i, y_j$, $i = 1 \ldots t$, $j = 1 \ldots m$, are maintained within a factor of $1 \pm \epsilon$. This readily yields (2). For (3) we observe that $2\langle xy \rangle = \|x\|^2 + \|y^2\| - \|x - y\|^2$ and thus if $\|\widehat{x}\|^2, \|\widehat{y}\|^2$ and $\|\widehat{x} - \widehat{y}\|^2$ are within $1 \pm \epsilon$ of $\|x\|^2, \|y\|^2$ and $\|x - y\|^2$, then (3) holds. $\square$

## 6    Conclusion

We have described three techniques for speeding up kernel methods through the use of randomization. While the discussion has focused on Kernel PCA, we feel that our techniques have potential for further development and empirical evaluation in a more general setting.

Indeed, the methods for sampling kernel expansions and for speeding up the kernel evaluation are universal; also, the Gram matrix sampling is readily applicable to any kernel technique based on the eigendecomposition of the Gram matrix [3]. Furthermore, it might enable us to speed up SVM training by sparsifying the Hessian and then applying a sparse QP solver, such as the ones described in [6, 9].

Our sampling and quantization techniques, both in training and classification, amount to repeatedly replacing single kernel evaluations with independent random variables that have appropriate expectations. Note, for example, that while we have represented the sampling of the kernel expansion as randomized rounding of coefficients, this rounding is also equivalent to the following process: consider each coefficients as is, but replace every kernel invocation $k(x, x_j)$ with an invocation of a randomized kernel function, distributed as

$$\widehat{k}_j(x, x_j) \ = \ \begin{cases} k(x, x_j)/|\alpha_j| & \text{with probability } s|\alpha_j| \\ \\ 0 & \text{otherwise.} \end{cases}$$

Similarly, the process of sampling in training can be thought of as replacing $k$ with

$$\widehat{k}(x_i, x_j) \ = \ \begin{cases} 0 & \text{with probability } 1 - 1/s \\ \\ sk(x_i, x_j) & \text{with probability } 1/s \ , \end{cases}$$

while an analogous randomized kernel is the obvious choice for quantization.

We feel that this approach suggests a notion of *randomized kernels*, wherein kernel evaluations are no longer considered as deterministic but inherently random, providing unbiased estimators for the corresponding inner products. Given bounds on the variance of these estimators, it seems that algorithms which reduce to computing weighted sums of kernel evaluations can exploit concentration of measure. Thus, randomized kernels appear promising as a general tool for speeding up kernel methods, warranting further investigation.

**Acknowledgments.** BS would like to thank Santosh Venkatesh for detailed discussions on sampling kernel expansions.

## Footnotes

[1]Our discussion applies equally well to Lanczos Iteration which, while often preferable, is a more complicated method. Here we focus on Orthogonal Iteration to simplify exposition.

## References

[1] D. Achlioptas, *Database-friendly random projections*, Proc. of the 20th Symposium on Principle of Database Systems (Santa Barbara, California), 2001, pp. 274–281.

[2] C. J. C. Burges, *Simplified support vector decision rules*, Proc. of the 13th International Conference on Machine Learning, Morgan Kaufmann, 1996, pp. 71–77.

[3] N. Cristianini, J. Shawe-Taylor, and H. Lodhi, *Latent semantic kernels*, Proc. of the 18th International Conference on Machine Learning, Morgan Kaufman, 2001.

[4] C. Davis and W. Kahan, *The rotation of eigenvectors by a perturbation 3*, SIAM Journal on Numerical Analysis **7** (1970), 1–46.

[5] Z. Füredi and J. Komlós, *The eigenvalues of random symmetric matrices*, Combinatorica **1** (1981), no. 3, 233–241.

[6] N. I. M. Gould, *An algorithm for large-scale quadratic programming*, IMA Journal of Numerical Analysis **11** (1991), no. 3, 299–324.

[7] W. Hoeffding, *Probability inequalities for sums of bounded random variables*, Journal of the American Statistical Association **58** (1963), 13–30.

[8] W. B. Johnson and J. Lindenstrauss, *Extensions of Lipschitz mappings into a Hilbert space*, Conference in modern analysis and probability (New Haven, Conn., 1982), American Mathematical Society, 1984, pp. 189–206.

[9] R. H. Nickel and J. W. Tolle, *A sparse sequential quadratic programming algorithm*, Journal of Optimization Theory and Applications **60** (1989), no. 3, 453–473.

[10] E. Osuna, R. Freund, and F. Girosi, *An improved training algorithm for support vector machines*, Neural Networks for Signal Processing VII, 1997, pp. 276–285.

[11] B. Schölkopf, A. J. Smola, and K.-R. Müller, *Nonlinear component analysis as a kernel eigenvalue problem*, Neural Computation **10** (1998), 1299–1319.

[12] A. J. Smola and B. Schölkopf, *Sparse greedy matrix approximation for machine learning*, Proc. of the 17th International Conference on Machine Learning, Morgan Kaufman, 2000, pp. 911–918.

[13] V. Vapnik, *The nature of statistical learning theory*, Springer, NY, 1995.

[14] C. K. I. Williams and M. Seeger, *Using the Nystrom method to speed up kernel machines*, Advances in Neural Information Processing Systems 2000, MIT Press, 2001.